# An Analog VLSI Model of Central Pattern Generation in the Leech

**Micah S. Siegel**[*]
Department of Electrical Engineering
Yale University
New Haven, CT 06520

## Abstract

I detail the design and construction of an analog VLSI model of the neural system responsible for swimming behaviors of the leech. Why the leech? The biological network is small and relatively well understood, and the silicon model can therefore span three levels of organization in the leech nervous system (neuron, ganglion, system); it represents one of the first comprehensive models of leech swimming operating in real-time. The circuit employs biophysically motivated analog neurons networked to form multiple biologically inspired silicon ganglia. These ganglia are coupled using known interganglionic connections. Thus the model retains the flavor of its biological counterpart, and though simplified, the output of the silicon circuit is similar to the output of the leech swim central pattern generator. The model operates on the same time- and spatial-scale as the leech nervous system and will provide an excellent platform with which to explore real-time adaptive locomotion in the leech and other "simple" invertebrate nervous systems.

## 1. INTRODUCTION

A Central Pattern Generator (CPG) is a network of neurons that generates rhythmic output in the absence of sensory input (Rowat and Selverston, 1991). It has been

[*] Present address: Micah Siegel, Computation and Neural Systems, Mail Stop 139-74 California Institute of Technology, Pasadena, CA 91125.

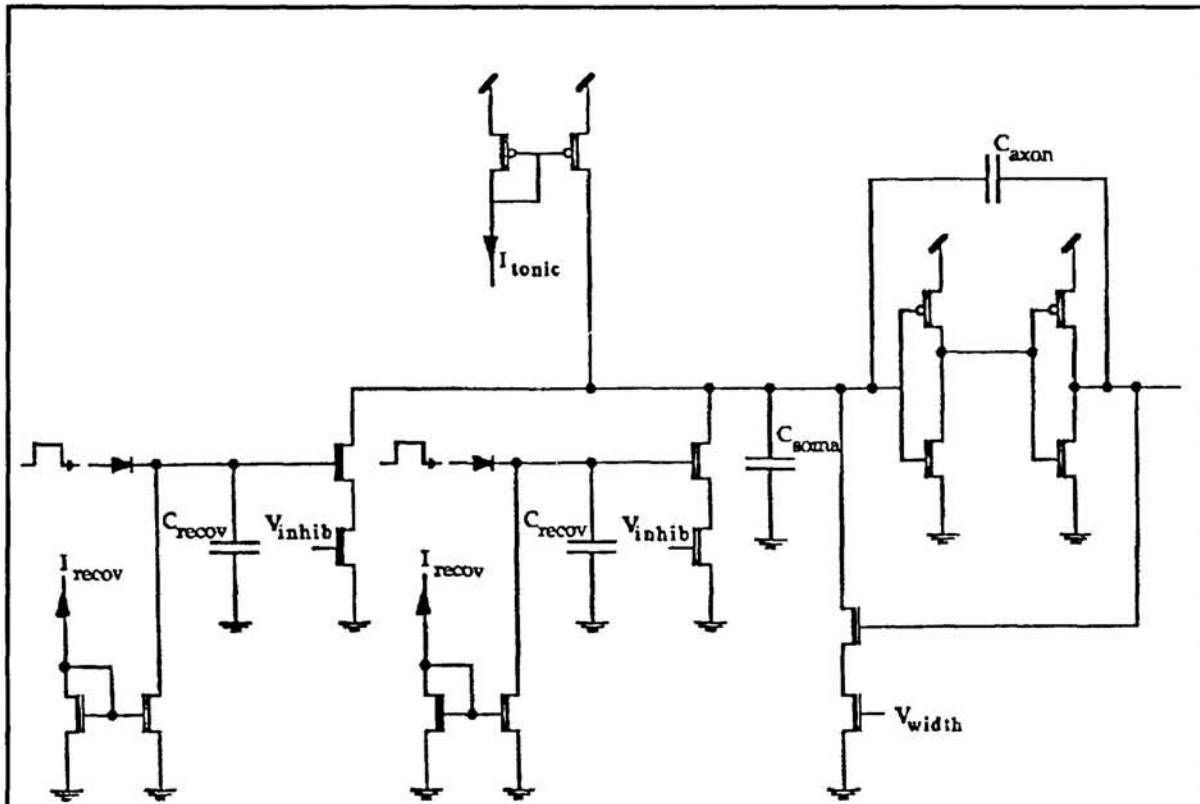

Figure 1. Silicon neuromime. The circuit includes tonic excitation, inhibitory synapses and an inhibitory recovery time. Note that there are two inhibitory synapses per device. $I_{ionic}$ sets the level of tonic excitatory input; $V_{inhib}$ sets the synaptic strength; $I_{recov}$ determines the inhibitory recovery time.

suggested that invertebrate central pattern generation may represent an excellent theatre within which to explore silicon implementations of adaptive neural systems: invertebrate CPG networks are orders of magnitude smaller than their vertebrate counterparts, much detailed information is available about them, and they guide behaviors that may be of technological interest (Ryckebusch et al., 1989). Furthermore, CPG networks are typically embedded in larger neural circuits and are integral to the neural correlates of adaptive behavior in many natural organisms (Friesen, 1989).

On strategy for modeling "simple" adaptive behaviors is first to evolve a biologically plausible framework within which to include increasingly more sophisticated and verisimilar adaptive mechanisms; because the model of leech swimming presented in this paper encompasses three levels of organization in the leech central nervous system, it may provide an ideal such structure with which to explore potentially useful adaptive mechanisms in the leech behavioral repertoire. Among others, these mechanisms include: habituation of the swim response (Debski and Friesen, 1985), the local bending reflex (Lockery and Kristan, 1990), and conditioned learning of the stepping and shortening behaviors (Sahley and Ready, 1988).

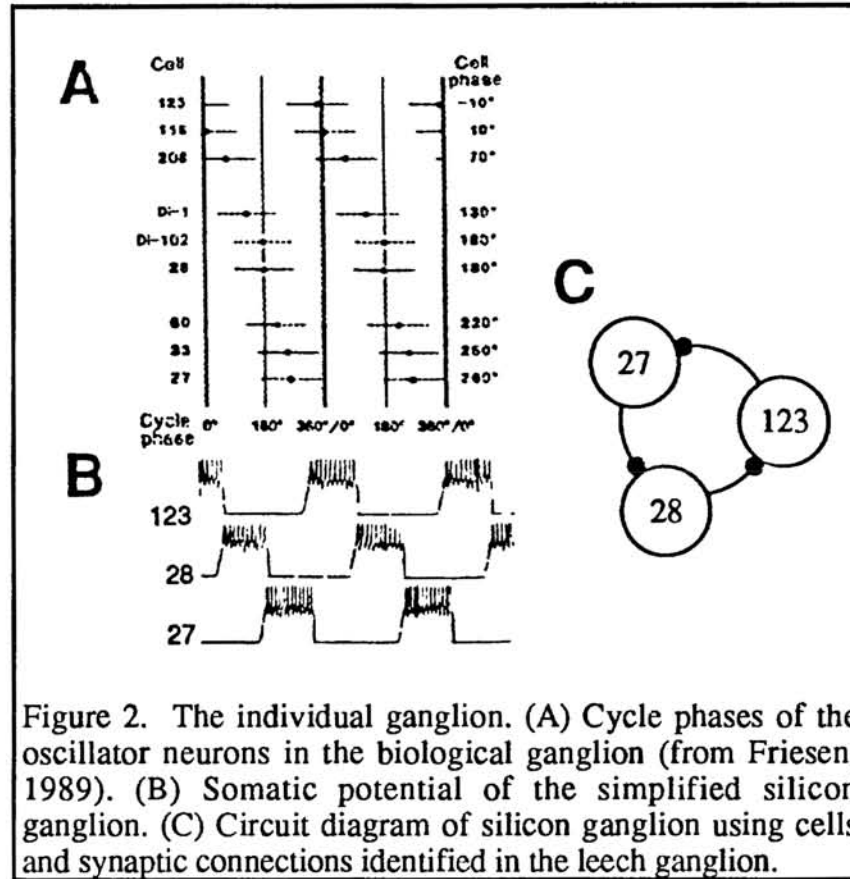

Figure 2. The individual ganglion. (A) Cycle phases of the oscillator neurons in the biological ganglion (from Friesen, 1989). (B) Somatic potential of the simplified silicon ganglion. (C) Circuit diagram of silicon ganglion using cells and synaptic connections identified in the leech ganglion.

## 2. LOCOMOTORY CPG IN THE LEECH

As a first step toward modeling a full repertoire of adaptive behavior in the medicinal leech (*Hirundo medicinalis*), I have designed, fabricated, and successfully tested an analog silicon model of one critical neural subsystem — the coupled oscillatory central pattern generation network responsible for swimming.  A leech swims by undulating its segmented body to form a rearward-progressing body wave.  This wave is analogous to the locomotory undulations of most elongated aquatic animals (e.g. fish), and some terrestrial amphibians and reptiles (including salamanders and snakes) (Friesen, 1989).  The moving crests and troughs in the body wave are produced by phase-delayed contractile rhythms of the dorsal and ventral body wall along successive segments (Stent and Kristan, 1981).  The interganglionic neural subsystem that subserves this behavior constitutes an important modeling platform because it guides locomotion in the leech over a wide range of frequencies and adapts to varying intrinsic and extrinsic conditions (Debski and Friesen, 1985).

In the medicinal leech, interneurons that coordinate the rearward-progressing swimming contractions undergo oscillations in membrane potential and fire impulses in bursts.  It appears that the oscillatory activity of these interneurons arises from a network rhythm that depends on synaptic interaction between neurons rather than from an endogenous polarization rhythm arising from inherently oscillatory membrane potentials in individual

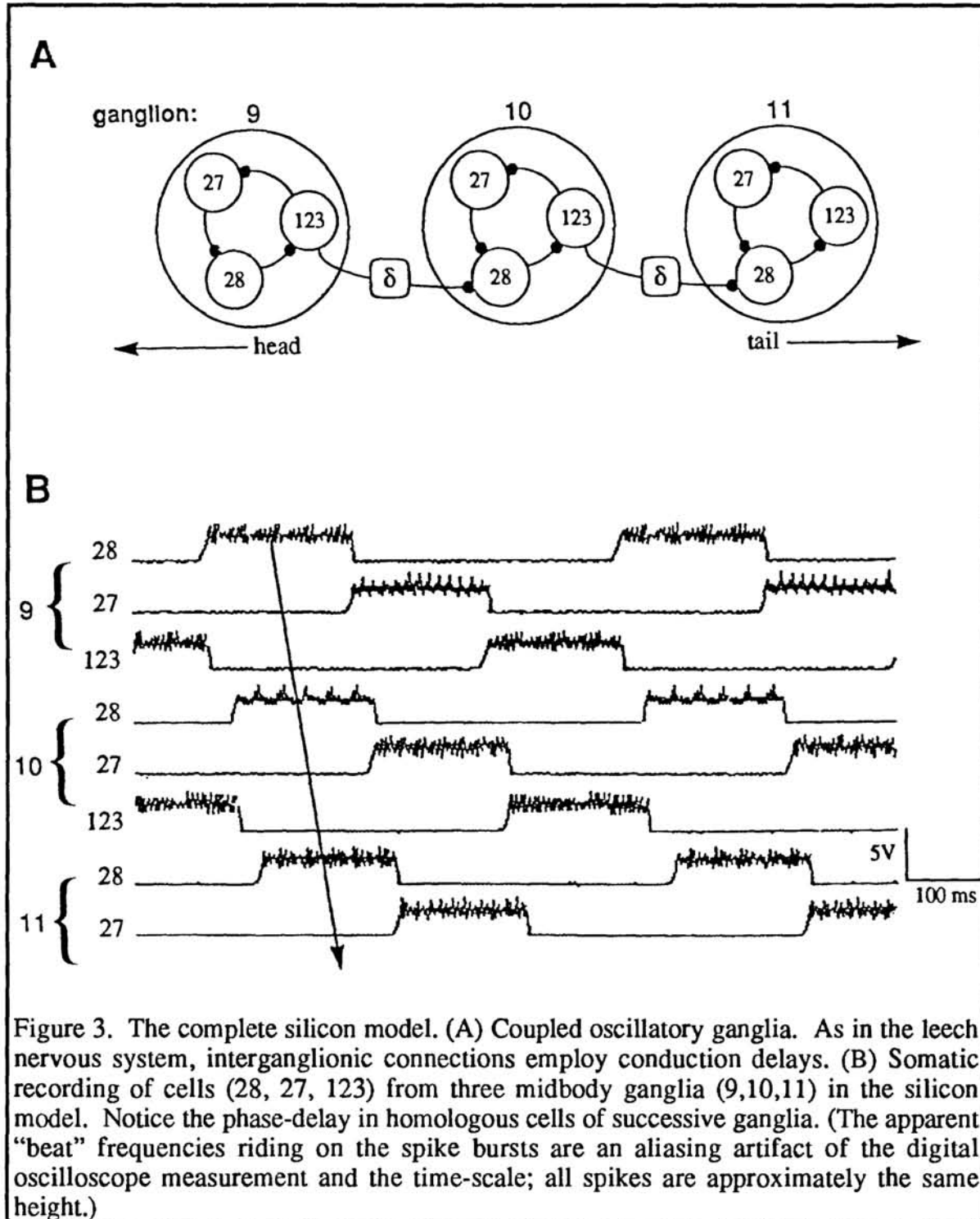

Figure 3. The complete silicon model. (A) Coupled oscillatory ganglia. As in the leech nervous system, interganglionic connections employ conduction delays. (B) Somatic recording of cells (28, 27, 123) from three midbody ganglia (9,10,11) in the silicon model. Notice the phase-delay in homologous cells of successive ganglia. (The apparent "beat" frequencies riding on the spike bursts are an aliasing artifact of the digital oscilloscope measurement and the time-scale; all spikes are approximately the same height.)

neurons (Friesen, 1989). The phases of the oscillatory interneurons form groups clustered about **three** phase points spaced equally around the activity cycle. To first approximation, all midbody ganglia of the leech nerve cord express an identical activity rhythm. However, activity in each ganglion is **phase-delayed** with respect to more anterior ganglia (Friesen, 1989); presumably this is responsible for the undulatory body wave characteristic of leech swimming.

## 3. THE SILICON MODEL

The silicon analog model employs biophysically realistic neural elements (neuromimes), connected into biologically realistic ganglion circuits. These ganglion circuits are coupled together using known interganglionic connections. This silicon model thus spans **three levels of organization** in the nervous system of the leech (neuron, ganglion, system), and represents one of the first comprehensive models of leech swimming (see also Friesen and Stent, 1977). The hope is that this model will provide a framework for the implementation of adaptive mechanisms related to undulatory locomotion in the leech and other invertebrates.

The building block of the model CPG is the analog neuromime (see figure 1); it exhibits many essential similarities to its biological counterpart. Like CPG interneurons in the leech swim system, the silicon neuromime integrates current across a somatic "capacitance" and uses positive feedback to generate action potentials whose frequency is determined by the magnitude of excitatory current input (Mead, 1989). In the leech swim system, nearly tonic excitatory input is transformed by a system of inhibition to produce the swim pattern (Friesen, 1989); adjustable **tonic excitation** is therefore included in the individual silicon neuromime.

**Inhibitory synapses** with adjustable weights are also implemented. Like its biological counterpart, the silicon neuromime includes a characteristic recovery time from inhibition. From theoretical and experimental studies, such inhibition recovery time is thought to play an important functional role in the interneurons that constitute the leech swim system (Friesen and Stent, 1977). Axonal delays have been demonstrated in the intersegmental interaction between ganglia in the leech. Similar axonal delays have been implemented in the silicon model using shifting delay lines.

The building block of the distributed model for the leech swim system is the ganglion. These biologically motivated silicon ganglia are constructed using only (though not all) identified cells and synaptic connections between cells in the biological system. Cells 27, 28, and 123 constitute a central inhibitory loop within each ganglion. Figure 2 exhibits the simplified diagram and the cycle phases of oscillatory interneurons in both the biological and the silicon ganglion. As in the leech ganglion, the phase relationships in the model ganglion fall into three groups, with cells 27, 28, and 123 participating each in the appropriate group of the oscillatory cycle. It is interesting that, though the silicon model captures the spirit of the tri-phasic output, the model is imprecise with respect to the exact phase locations of cells 27, 28, and 123 within their respective groups. This discrepancy between the silicon model and the biological system may point to the significance of other swim interneurons for swim pattern generation in the leech. Undoubtedly, the additional oscillatory interneurons sculpt this tri-phasic output significantly.

The silicon model of coupled successive segments in the leech is implemented using these silicon neurons and biologically motivated ganglia. The model employs interganglionic connections known to exist in the biological system and generates qualitatively similar output at the same time-scale as the leech system. It appears in the leech that synchronization between ganglia is governed by the interganglionic synaptic interaction of interneurons involved in the oscillatory pattern rather than by autonomous

coordinating neurons (Friesen, 1989). In the silicon model, interganglionic interaction is represented by a projection from more anterior cell 123 to more posterior cell 28; this

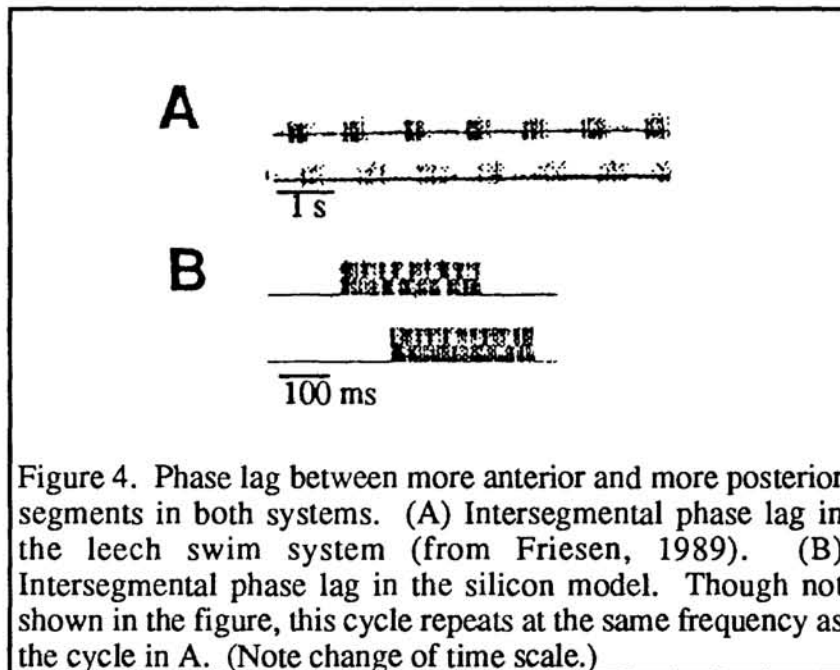

Figure 4. Phase lag between more anterior and more posterior segments in both systems. (A) Intersegmental phase lag in the leech swim system (from Friesen, 1989). (B) Intersegmental phase lag in the silicon model. Though not shown in the figure, this cycle repeats at the same frequency as the cycle in A. (Note change of time scale.)

projection is also observed between cells 123 and 28 of successive ganglia in the leech (Friesen, 1989), however it is by no means the only such interganglionic connection. In addition, the biological system utilizes conduction delays in its interganglionic projections; each of these is modeled in the silicon system by a delay line (Friesen and Stent, 1977) analogous to an active cable with adjustable propagation speed. Figure 3 demonstrates the silicon model of three coupled ganglia with transmission delays. Notice that neuromimes in each successive ganglion are phase-delayed from homologous neuromimes in more anterior ganglia. Figure 4 shows this phase delay more explicitly.

## 4. DISCUSSION

The analog silicon model of central pattern generation in the leech successfully captures design principles from three levels of organization in the leech nervous system and has been tested over a wide range of network parameter values. It operates on the same time-scale as its biological counterpart and gives rise to ganglionic activity that is qualitatively similar to activity in the leech ganglion. Furthermore, it maintains biologically plausible phase relationship between homologous elements of successive ganglia. The design of the silicon model is intentionally compatible with analog Very Large Scale Integration (VLSI) technology, making its integrated spatial-scale close to that of the leech nervous system. It is interesting that this highly simplified model captures qualitatively the output both within and between ganglia of the leech; it may be illuminating to explore the functional significance of other swim interneurons by their inclusion in similar silicon networks. The current model provides an important platform for future implementations of invertebrate adaptive behaviors, especially those behaviors related to swim and other locomotory pattern generation. The hope is that such behaviors

can be evolved incrementally using neuromime models of identified adaptive interneurons to modulate the swim central pattern generating network.

## Acknowledgments

I would like to thank the department of Electrical Engineering at Yale University for encouraging and generously supporting independent undergraduate research.

## References

Rowat, P.F. and Selverston, A.I. (1991). *Network*, 2, 17-41.

Ryckebusch, S., Bower, J.M., Mead, C., (1989). In D.Touretzky (ed.), *Advances in Neural Information Processing Systems*, 384-393. San Mateo, CA: Morgan Kaufmann.

Friesen, W.O. (1989). In J. Jacklet (ed), *Neuronal and Cellular Oscillators*, 269-316. New York: Marcel Dekker.

Debski, E.A. and Friesen, W.O. (1985). *Journal of Experimental Biology*, 116, 169-188.

Lockery, S.R. and Kristan, W.B. (1990). *Journal of Neuroscience*, 10(6), 1811-1815.

Sahley, C.L. and Ready, D.F. (1988). *Journal of Neuroscience*, 8(12), 4612-4620.

Stent, G.S. and Kristan, W.B. (1981). In K.Muller, J Nicholls, and G. Stent (eds), *Neurobiology of the Leech*, 113-146. Cold Spring Harbor: Cold Spring Harbor Laboratory.

Mead, C.A. (1989). *Analog VLSI and Neural Systems*, Reading, MA: Addison-Wesley.

Friesen, W.O. and Stent, G.S. (1977). *Biological Cybernetics*, 28, 27-40.